# A Mixture Model System for Medical and Machine Diagnosis

**Magnus Stensmo**          **Terrence J. Sejnowski**

Computational Neurobiology Laboratory
The Salk Institute for Biological Studies
10010 North Torrey Pines Road
La Jolla, CA 92037, U.S.A.
{magnus,terry}@salk.edu

## Abstract

Diagnosis of human disease or machine fault is a missing data problem since many variables are initially unknown. Additional information needs to be obtained. The joint probability distribution of the data can be used to solve this problem. We model this with mixture models whose parameters are estimated by the EM algorithm. This gives the benefit that missing data in the database itself can also be handled correctly. The request for new information to refine the diagnosis is performed using the maximum utility principle. Since the system is based on learning it is domain independent and less labor intensive than expert systems or probabilistic networks. An example using a heart disease database is presented.

## 1 INTRODUCTION

Diagnosis is the process of identifying diseases in patients or disorders in machines by considering history, symptoms and other signs through examination. Diagnosis is a common and important problem that has proven hard to automate and formalize. A procedural description is often hard to attain since experts do not know exactly how they solve a problem.

In this paper we use the information about a specific problem that exists in a database

of cases. The disorders or diseases are determined by variables from observations and the goal is to find the probability distribution over the disorders, conditioned on what has been observed. The diagnosis is strong when one or a few of the possible outcomes are differentiated from the others. More information is needed if it is inconclusive. Initially there are only a few clues and the rest of the variables are unknown. Additional information is obtained by asking questions and doing tests. Since tests may be dangerous, time consuming and expensive, it is generally not possible or desirable to find the answer to every question. Unnecessary tests should be avoided.

There have been many attempts to automate diagnosis. Early work [Ledley & Lusted, 1959] realized that the problem is not always tractable due to the large number of influences that can exist between symptoms and diseases. Expert systems, e.g. the INTERNIST system for internal medicine [Miller et al., 1982], have rule-bases which are very hard and time consuming to build. Inconsistencies may arise when new rules are added to an existing database. There is also a strong domain dependence so knowledge bases can rarely be reused for new applications.

Bayesian or probabilistic networks [Pearl, 1988] are a way to model a joint probability distribution by factoring using the chain rule in probability theory. Although the models are very powerful when built, there are presently no general learning methods for their construction. A considerable effort is needed. In the Pathfinder system for lymph node pathology [Heckerman et al., 1992] about 14,000 conditional probabilities had to be assessed by an expert pathologist. It is inevitable that errors will occur when such large numbers of manual assessments are involved.

Approaches to diagnosis that are based on domain-independent machine learning alleviate some of the problems with knowledge engineering. For decision trees [Quinlan, 1986], a piece of information can only be used if the appropriate question comes up when traversing the tree. This means that irrelevant questions can not be avoided. Feedforward multilayer perceptrons for diagnosis [Baxt, 1990] can classify very well, but they need full information about a case. None of these these methods have adequate ways to handle missing data during learning or classification.

The exponentially growing number of probabilities involved can make exact diagnosis intractable. Simple approximations such as independence between all variables and conditional independence given the disease (naive Bayes) introduce errors since there usually are dependencies between the symptoms. Even though systems based on these assumptions work surprisingly well, correct diagnosis is not guaranteed. This paper will avoid these assumptions by using mixture models.

## 2   MIXTURE MODELS

Diagnosis can be formulated as a probability estimation problem with missing inputs. The probabilities of the disorders are conditioned on what has currently been observed. If we model the *joint probability distribution* it is easy to marginalize to get any conditional probability. This is necessary in order to be able to handle missing data in a principled way [Ahmad & Tresp, 1993]. Using *mixture models* [McLachlan & Basford, 1988], a simple closed form solution to optimal regression with missing data can be formulated. The EM algorithm, a method from parametric statistics for parameter estimation, is especially interesting in this context since it can also be formulated to handle missing data in the

training examples [Dempster et al., 1977; Ghahramani & Jordan, 1994].

## 2.1 THE EM ALGORITHM

The data underlying the model is assumed to be a set of $N$ $D$-dimensional vectors $\boldsymbol{X} = \{\boldsymbol{x}_1, \ldots, \boldsymbol{x}_N\}$. Each data point is assumed to have been generated independently from a *mixture density* with $M$ components

$$p(\boldsymbol{x}) = \sum_{j=1}^{M} p(\boldsymbol{x}, \omega_j; \theta_j) = \sum_{j=1}^{M} p(\omega_j)p(\boldsymbol{x}|\omega_j; \theta_j), \tag{1}$$

where each mixture component is denoted by $\omega_j$. $p(\omega_j)$, the a priori probability for mixture $\omega_j$, and $\boldsymbol{\theta} = (\theta_1, \ldots, \theta_M)$ are the model parameters.

To estimate the parameters for the different mixtures so that it is likely that the linear combination of them generated the set of data points, we use *maximum likelihood estimation*. A good method is the iterative *Expectation-Maximization*, or *EM*, algorithm [Dempster et al., 1977].

Two steps are repeated. First a likelihood is formulated and its expectation is computed in the *E-step*. For the type of models that we will use, this step will calculate the probability that a certain mixture component generated the data point in question. The second step is the *M-step* where the parameters that maximize the expectation are found. This can be found analytically for models that can be written in an exponential form, e.g. Gaussian functions. Equations can be derived for both batch and on-line learning. Update equations for Gaussian distributions with and without missing data will be given here, other distributions are possible, e.g. binomial or multinomial [Stensmo & Sejnowski, 1994]. Details and derivations can be found in [Dempster et al., 1977; Nowlan, 1991; Ghahramani & Jordan, 1994; Stensmo & Sejnowski, 1994].

From (1) we form the log likelihood of the data

$$L(\boldsymbol{\theta}|\boldsymbol{X}) = \sum_{i=1}^{N} \log p(\boldsymbol{x}_i; \theta_j) = \sum_{i=1}^{N} \log \sum_{j=1}^{M} p(\omega_j)p(\boldsymbol{x}_i|\omega_j; \theta_j).$$

There is unfortunately no analytic solution to the logarithm of the sum in the right hand side of the equation. However, if we were to know which of the mixtures generated which data point we could compute it. The EM algorithm solves this by introducing a set of binary indicator variables $\boldsymbol{Z} = \{z_{ij}\}$. $z_{ij} = 1$ if and only if the data point $\boldsymbol{x}_i$ was generated by mixture component $j$. The log likelihood can then be manipulated to a form that does not contain the log of a sum.

The expectation of $\boldsymbol{z}_i$ using the current parameter values $\boldsymbol{\theta}_k$ is used since $\boldsymbol{z}_i$ is not known directly. This is the *E-step* of the EM algorithm. The expected value is then maximized in the *M-step*. The two steps are iterated until convergence. The likelihood will never decrease after an iteration [Dempster et al., 1977]. Convergence is fast compared to gradient descent.

One of the main motivations for the EM-algorithm was to be able to handle missing values for variables in a data set in a principled way. In the complete data case we introduced missing indicator variables that helped us solve the problem. With missing data we add the missing components to the $\boldsymbol{Z}$ already missing [Dempster et al., 1977; Ghahramani & Jordan, 1994].

## 2.2  GAUSSIAN MIXTURES

We specialize here the EM algorithm to the case where the mixture components are radial Gaussian distributions. For mixture component $j$ with mean $\mu_j$ and covariance matrix $\Sigma_j$ this is

$$\mathrm{p}(\boldsymbol{x}|\omega_j) = G_j(\boldsymbol{x}) = \left(\frac{1}{\sqrt{2\pi}}\right)^D |\Sigma_j|^{-\frac{1}{2}} \exp\left[-\frac{1}{2}(\boldsymbol{x}-\mu_j)^T \Sigma_j^{-1}(\boldsymbol{x}-\mu_j)\right].$$

The form of the covariance matrix is often constrained to be diagonal or to have the same values on the diagonal, $\Sigma_j = \sigma_j^2 I$. This corresponds to axis-parallel oval-shaped and radially symmetric Gaussians, respectively. Radial and diagonal basis functions can function well in applications [Nowlan, 1991], since several Gaussians together can form complex shapes in the space. With fewer parameters over-fitting is minimized. In the radial case, with variance $\sigma_j^2$

$$G_j(\boldsymbol{x}) = \left(\frac{1}{\sqrt{2\pi}\sigma_j}\right)^D \exp\left[-\frac{\|\boldsymbol{x}-\mu_j\|^2}{2\sigma_j^2}\right].$$

In the E-step the expected value of the likelihood is computed. For the Gaussian case this becomes the probability that Gaussian $j$ generated the data point

$$p_j(\boldsymbol{x}) = \frac{\mathrm{p}(\omega_j)G_j(\boldsymbol{x})}{\sum_{k=1}^M \mathrm{p}(\omega_k)G_k(\boldsymbol{x})}.$$

The M-step finds the parameters that maximize the likelihood from the E-step. For complete data the new estimates are

$$\hat{p}(\omega_j) \longleftarrow \frac{S_j}{N}, \qquad \hat{\mu}_j \longleftarrow \frac{1}{S_j}\sum_{i=1}^N p_j(\boldsymbol{x}_i)\boldsymbol{x}_i, \qquad (2)$$

$$\hat{\sigma}_j^2 \longleftarrow \frac{1}{DS_j}\sum_{i=1}^N p_j(\boldsymbol{x}_i)\|\boldsymbol{x}_i - \hat{\mu}_j\|^2, \qquad \text{where } S_j = \sum_{i=1}^N p_j(\boldsymbol{x}_i).$$

When input variables are missing the $G_j(\boldsymbol{x})$ is only evaluated over the set of observed dimensions $O$. Missing (unobserved) dimensions are denoted by $U$. The update equation for $\hat{p}(\omega_j)$ is unchanged. To estimate $\hat{\mu}_j$ we set $\boldsymbol{x}_i^U = \hat{\mu}_j^U$ and use (2). The variance becomes

$$\hat{\sigma}_j^2 \longleftarrow \frac{1}{DS_j}\sum_{i=1}^N p_j^O(\boldsymbol{x}_i)\left[\|\boldsymbol{x}_i^O - \hat{\mu}_j^O\|^2 + |U|\hat{\sigma}_j^2\right].$$

A least squares regression was used to fill in missing data values during classification. For missing variables and Gaussian mixtures this becomes the same approach used by [Ahmad & Tresp, 1993]. The result of the regression when the outcome variables are missing is a probability distribution over the disorders. This can be reduced to a classification for comparison with other systems by picking the outcome with the maximum of the estimated probabilities.

## 3 REQUESTING MORE INFORMATION

During the diagnosis process, the outcome probabilities are refined at each step based on newly acquired knowledge. It it important to select the questions that lead to the minimal number of necessary tests. There is generally a cost associated with each test and the goal is to minimize the total cost. Early work on automated diagnosis [Ledley & Lusted, 1959] acknowledged the problem of asking as few questions as possible and suggested the use of *decision analysis* for the solution. An important idea from the field of decision theory is the *maximum expected utility principle* [von Neuman & Morgenstern, 1947]: A decision maker should always choose the alternative that maximizes some expected utility of the decision. For diagnosis it is the cost of misclassification. Each pair of outcomes has a utility $u(x, y)$ when the correct diagnosis is $x$ but $y$ has been incorrectly determined. The expectation can be computed when we know the probabilities of the outcomes.

The utility values have to be assessed manually in what can be a lengthy and complicated process. For this reason a simplification of this function has been suggested by [Heckerman et al., 1992]: The utility $u(x, y)$ is 1 when both $x$ and $y$ are benign or both are malign, and 0 otherwise. This simplification has been found to work well in practice. Another complication with maximum expected utility principle can also make it intractable. In the ideal case we would evaluate every possible sequence of future choices to see which is the best. Since the size of the search tree of possibilities grows exponentially this is often not possible. A simplification is to look ahead only one or a few steps at a time. This nearsighted or *myopic* approach has been tested in practice with good results [Gorry & Barnett, 1967; Heckerman et al., 1992].

## 4 THE DIAGNOSIS SYSTEM

The system we have developed has two phases. First there is a learning phase where a probabilistic model is built. This model is then used for inference in the diagnosis phase.

In the learning phase, the joint probability distribution of the data is modeled using mixture models. Parameters are determined from a database of cases by the EM algorithm. The $k$-means algorithm is used for initialization. Input and output variables for each case are combined into one vector per case to form the set of training patterns. The outcomes and other nominal variables are coded as *1 of N*. Continuous variables are interval coded.

In the diagnosis phase, myopic one-step look-ahead was used and utilities were simplified as above. The following steps were performed:

1. Initial observations were entered.

2. Conditional expectation regression was used to fill in unknown variables.

3. The maximum expected utility principle was used to recommend the next observation to make. Stop if nothing would be gained by further observations.

4. The user was asked to determine the correct value for the recommended observation. Any other observations could be made, instead of or in addition to this.

5. Continue with step 2.

Table 1: The Cleveland Heart Disease database.

|    | Observation | Description                                 | Values                  |
|----|-------------|---------------------------------------------|-------------------------|
| 1  | age         | Age in years                                | continuous              |
| 2  | sex         | Sex of subject                              | male/female             |
| 3  | cp          | Chest pain                                  | four types              |
| 4  | trestbps    | Resting blood pressure                      | continuous              |
| 5  | chol        | Serum cholesterol                           | continuous              |
| 6  | fbs         | Fasting blood sugar                         | lt or gt 120 mg/dl      |
| 7  | restecg     | Resting electrocardiogr.                    | five values             |
| 8  | thalach     | Max heart rate achieved                     | continuous              |
| 9  | exang       | Exercise induced angina                     | yes/no                  |
| 10 | oldpeak     | ST depr. induced by exercise relative to rest | continuous           |
| 11 | slope       | Slope of peak exercise ST segment           | up/flat/down            |
| 12 | ca          | # major vess. col. flourosc.                | 0-3                     |
| 13 | thal        | Defect type                                 | normal/fixed/reversible |
|    | **Disorder** | **Description**                            | **Values**              |
| 14 | num         | Heart disease                               | Not present/4 types     |

## 5  EXAMPLE

The Cleveland heart disease data set from UC, Irvine has been used to test the system. It contains 303 examples of four types of heart disease and its absence. There are thirteen continuous- or nominally-valued variables (Table 1). The continuous variables were interval coded with one unit per standard deviation away from the mean value. This was chosen since they were approximately normally distributed. Nominal variables were coded with one unit per value. In total the 14 variables were coded with 55 units. The EM steps were repeated until convergence (60–150 iterations). A varying number of mixture components (20–120) were tried.

Previously reported results have used only presence or absence of the heart disease. The best of these has been a classification rate of 78.9% using a system that incrementally built prototypes [Gennari et al., 1989]. We have obtained 78.6% correct classification with 60 radial Gaussian mixtures as described above. Performance increased with the number of mixture components. It was not sensitive to a varying number of mixture components during training unless there were too few of them. Previous investigators have pointed out that there is not enough information in the thirteen variables in this data set to reach 100% [Gennari et al., 1989].

An annotated transcript of a diagnosis session is shown in Figure 1.

## 6  CONCLUSIONS AND FURTHER WORK

Several properties of this model remain to be investigated. It should be tested on several more databases. Unfortunately databases are typically proprietary and difficult to obtain. Future prospects for medical databases should be good since some hospitals are now using computerized record systems instead of traditional paper-based. It should be fairly easy to

The leftmost number of the five numbers in a line is the estimated probability for no heart disease, followed by the probabilities for the four types of heart disease. The entropy, defined as $-\sum_i p_i \log p_i$, of the diagnoses are given at the same time as a measure of how decisive the current conclusion is. A completely determined diagnosis has entropy 0. Initially all of the variables are unknown and starting diagnoses are the unconditional prior probabilities.

```
Disorders (entropy = 1.85):
  0.541254  0.181518  0.118812  0.115512  0.042904
What is cp ? 3
```

The first question is *chest pain*, and the answer changes the estimated probabilities. This variable is continuous. The answer is to be interpreted how far from the mean the observation is in standard deviations. As the decision becomes more conclusive, the entropy decreases.

```
Disorders (entropy = 0.69):
  0.888209  0.060963  0.017322  0.021657  0.011848
What is age ? 0
```

```
Disorders (entropy = 0.57):
  0.91307619  0.00081289  0.02495360  0.03832095  0.02283637
What is oldpeak ? -2
```

```
Disorders (entropy = 0.38):
  0.94438718  0.00089016  0.02539957  0.02691099  0.00241210
What is chol ?  -1
```

```
Disorders (entropy = 0.11):
  0.98848758  0.00028553  0.00321580  0.00507073  0.00294036
```

We have now determined that the probability of no heart disease in this case is 98.8%. The remaining 0.2% is spread out over the other possibilities.

Figure 1: Diagnosis example.

generate data for machine diagnosis.

An alternative way to choose a new question is to evaluate the variance change in the output variables when a variable is changed from missing to observed. The idea is that a variable known with certainty has zero variance. The variable with the largest resulting *conditional variance* could be selected as the query, similar to [Cohn et al., 1995].

One important aspect of automated diagnosis is the accompanying explanation for the conclusion, a factor that is important for user acceptance. Since the basis functions have local support and since we have estimates for the probability of each basis function having generated the observed data, explanations for the conclusions could be generated.

Instead of using the simplified utilities with values 0 and 1 for the expected utility calculations they could be learned by reinforcement learning. A trained expert would evaluate the quality of the diagnosis performed by the system, followed by adjustment of the utilities. The 0 and 1 values can be used as starting values.

## Acknowledgements

The heart disease database is from the University of California, Irvine Repository of Machine Learning Databases and originates from R. Detrano, Cleveland Clinic Foundation. Peter Dayan provided helpful comments on an earlier version of this paper.

## References

Ahmad, S. & Tresp, V. (1993). Some solutions to the missing feature problem in vision. In *Advances in Neural Information Processing Systems*, vol. 5, pp 393–400. Morgan Kaufmann, San Mateo, CA.

Baxt, W. (1990). Use of an artificial neural network for data analysis in clinical decision-making: The diagnosis of acute coronary occlusion. *Neural Computation*, **2(4)**, 480–489.

Cohn, D. A., Ghahramani, Z. & Jordan, M. I. (1995). Active learning with statistical models. In *Advances in Neural Information Processing Systems*, vol. 7. Morgan Kaufmann, San Mateo, CA.

Dempster, A., Laird, N. & Rubin, D. (1977). Maximum likelihood from incomplete data via the EM algorithm. *Journal of the Royal Statistical Society, Series, B.*, **39**, 1–38.

Gennari, J., Langley, P. & Fisher, D. (1989). Models of incremental concept formation. *Artificial Intelligence*, **40**, 11–62.

Ghahramani, Z. & Jordan, M. (1994). Supervised learning from incomplete data via an EM approach. In *Advances in Neural Information Processing Systems*, vol. 6, pp 120–127. Morgan Kaufmann, San Mateo, CA.

Gorry, G. A. & Barnett, G. O. (1967). Experience with a model of sequential diagnosis. *Computers and Biomedical Research*, **1**, 490–507.

Heckerman, D., Horvitz, E. & Nathwani, B. (1992). Toward normative expert systems: Part I. The Pathfinder project. *Methods of Information in Medicine*, **31**, 90–105.

Ledley, R. S. & Lusted, L. B. (1959). Reasoning foundations of medical diagnosis. *Science*, **130(3366)**, 9–21.

McLachlan, G. J. & Basford, K. E. (1988). *Mixture Models: Inference and Applications to Clustering*. Marcel Dekker, Inc., New York, NY.

Miller, R. A., Pople, H. E. & Myers, J. D. (1982). Internist-1: An experimental computer-based diagnostic consultant for general internal medicine. *New England Journal of Medicine*, **307**, 468–476.

Nowlan, S. J. (1991). *Soft Competitive Adaptation: Neural Network Learning Algorithms based on Fitting Statistical Mixtures*. PhD thesis, School of Computer Science, Carnegie Mellon University, Pittsburgh, PA.

Pearl, J. (1988). *Probabilistic Reasoning in Intelligent Systems: Networks of Plausible Inference*. Morgan Kaufmann, San Mateo, CA.

Quinlan, J. R. (1986). Induction of decision trees. *Machine Learning*, **1**, 81–106.

Stensmo, M. & Sejnowski, T. J. (1994). A mixture model diagnosis system. Tech. Rep. INC-9401, Institute for Neural Computation, University of California, San Diego.

von Neuman, J. & Morgenstern, O. (1947). *Theory of Games and Economic Behavior*. Princeton University Press, Princeton, NJ.
